# Markov Random Fields Can Bridge Levels of Abstraction

**Paul R. Cooper**
Institute for the Learning Sciences
Northwestern University
Evanston, IL
cooper@ils.nwu.edu

**Peter N. Prokopowicz**
Institute for the Learning Sciences
Northwestern University
Evanston, IL
prokopowicz@ils.nwu.edu

## Abstract

Network vision systems must make inferences from evidential information across levels of representational abstraction, from low level invariants, through intermediate scene segments, to high level behaviorally relevant object descriptions. This paper shows that such networks can be realized as Markov Random Fields (MRFs). We show first how to construct an MRF functionally equivalent to a Hough transform parameter network, thus establishing a principled probabilistic basis for visual networks. Second, we show that these MRF parameter networks are more capable and flexible than traditional methods. In particular, they have a well-defined probabilistic interpretation, intrinsically incorporate feedback, and offer richer representations and decision capabilities.

## 1 INTRODUCTION

The nature of the vision problem dictates that neural networks for vision must make inferences from evidential information across levels of representational abstraction. For example, local image evidence about edges might be used to determine the occluding boundary of an object in a scene. This paper demonstrates that parameter networks [Ballard, 1984], which use voting to bridge levels of abstraction, can be realized with Markov Random Fields (MRFs).

We show two main results. First, an MRF is constructed with functionality formally equivalent to that of a parameter net based on the Hough transform. Establishing

this equivalence provides a sound probabilistic foundation for neural networks for vision. This is particularly important given the fundamentally evidential nature of the vision problem.

Second, we show that parameter networks constructed from MRFs offer a more flexible and capable framework for intermediate vision than traditional feedforward parameter networks with threshold decision making. In particular, MRF parameter nets offer a richer representational framework, the potential for more complex decision surfaces, an integral treatment of feedback, and probabilistically justified decision and training procedures. Implementation experiments demonstrate these features.

Together, these results establish a basis for the construction of integrated network vision systems with a single well-defined representation and control structure that intrinsically incorporates feedback.

## 2   BACKGROUND

### 2.1   HOUGH TRANSFORM AND PARAMETER NETS

One approach to bridging levels of abstraction in vision is to combine local, highly variable evidence into segments which can be described compactly by their parameters. The Hough transform offers one method for obtaining these high-level parameters. Parameter networks implement the Hough transform in a parallel feedforward network. The central idea is voting: local low-level evidence cast votes via the network for compatible higher-level parameterized hypotheses. The classic Hough example finds lines from edges. Here local evidence about the direction and magnitude of image contrast is combined to extract the parameters of lines (e.g. slope-intercept), which are more useful scene segments. The Hough transform is widely used in computer vision (e.g. [Bolle *et al.*, 1988]) to bridge levels of abstraction.

### 2.2   MARKOV RANDOM FIELDS

Markov Random Fields offer a formal foundation for networks [Geman and Geman, 1984] similar to that of the Boltzmann machine. MRFs define a prior joint probability distribution over a set **X** of discrete random variables. The possible values for the variables can be interpreted as possible local features or hypotheses. Each variable is associated with a node $S$ in an undirected graph (or network), and can be written as $X_s$. An assignment of values to all the variables in the field is called a configuration, and is denoted $\omega$; an assignment of a single variable is denoted $\omega_s$. Each fully-connected neighborhood $C$ in a configuration of the field has a weight, or clique potential, $V_c$.

We are interested in the probability distributions $P$ over the random field **X**. Markov Random Fields have a locality property:

$$P(X_s = \omega_s | X_r = \omega_r, r \in S, r \neq s) = P(X_s = \omega_s | X_r = \omega_r, r \in N_s) \qquad (1)$$

that says roughly that the state of site is dependent only upon the state of its neighbors ($N_s$). MRFs can also be characterized in terms of an energy function $U$

with a Gibb's distribution:

$$P(\omega) = \frac{e^{-U(\omega)/T}}{Z} \qquad (2)$$

where $T$ is the temperature, and $Z$ is a normalizing constant.

If we are interested only in the prior distribution $P(\omega)$, the energy function $U$ is defined as:

$$U(\omega) = \sum_{c \in C} V_c(\omega) \qquad (3)$$

where $C$ is the set of cliques defined by the neighborhood graph, and the $V_c$ are the clique potentials. Specifying the clique potentials thus provides a convenient way to specify the global joint prior probability distribution $P$, i.e. to encode prior domain knowledge about plausible structures.

Suppose we are instead interested in the distribution $P(\omega|O)$ on the field after an observation $O$, where an observation constitutes a combination of spatially distinct observations at each local site. The evidence from an observation at a site is denoted $P(O_s|\omega_s)$ and is called a likelihood. Assuming likelihoods are local and spatially distinct, it is reasonable to assume that they are conditionally independent. Then, with Bayes' Rule we can derive:

$$U(\omega|O) = \sum_{c \in C} V_c(\omega) - T \sum_{s \in S} \log P(O_s|\omega_s) \qquad (4)$$

The MRF definition, together with evidence from the current problem, leaves a probability distribution over all possible configurations. An algorithm is then used to find a solution, normally the configuration of maximal probability, or equivalently, minimal energy as expressed in equation 4. The problem of minimizing non-convex energy functions, especially those with many local minima, has been the subject of intense scrutiny recently (e.g. [Kirkpatrick et al., 1983; Hopfield and Tank, 1985]). In this paper we focus on developing MRF representations wherein the minimum energy configuration defines a desirable goal, not on methods of finding the minimum. In our experiments have have used the deterministic Highest Confidence First (HCF) algorithm [Chou and Brown, 1990].

MRFs have been widely used in computer vision applications, including image restoration, segmentation, and depth reconstruction [Geman and Geman, 1984; Marroquin, 1985; Chellapa and Jain, 1991]. All these applications involve flat representations at a single level of abstraction. A novel aspect of our work is the hierarchical framework which explicitly represents visual entities at different levels of abstraction, so that these higher-order entities can serve as an interpretation of the data as well as play a role in further constraint satisfaction at even higher levels.

## 3   CONSTRUCTING MRFS EQUIVALENT TO PARAMETER NETWORKS

Here we define a Markov Random Field that computes a Hough transform; i.e. it detects higher-order features by tallying weighted votes from low-level image components and thresholding the sum. The MRF has one discrete variable for

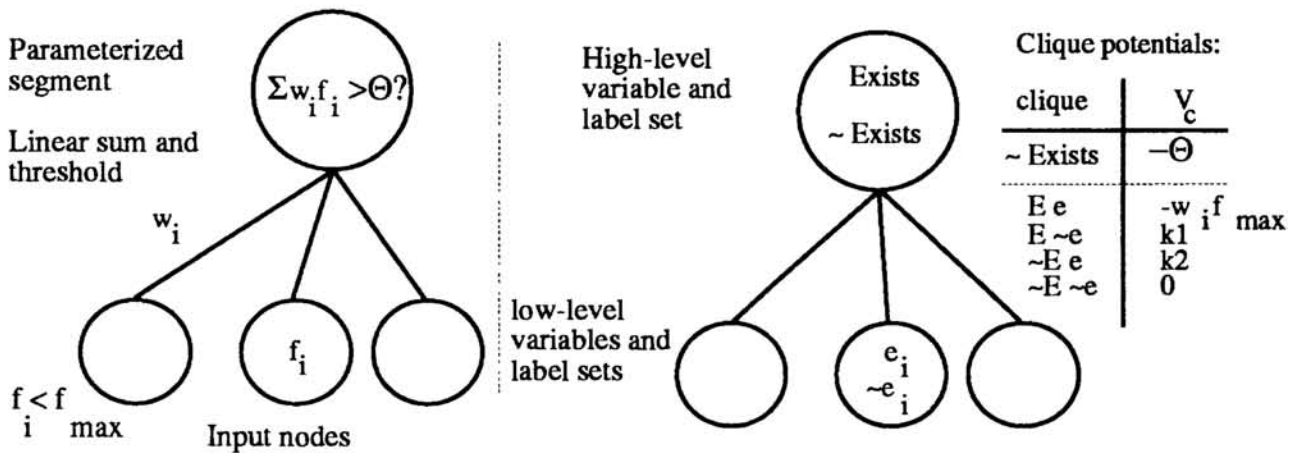

Figure 1: Left: Hough-transform parameter net. Input determines confidence $f_i$ in each low-level feature; these confidences are weighted ($w_i$), summed, and thresholded. Right: Equivalent MRF. Circles show variables with possible labels and non-zero unary clique potentials; lines show neighborhoods; potentials are for the four labellings of the binary cliques.

the higher-order feature, whose possible values are *exists* and *doesn't exist* and one discrete variable for each voting element, with the same two possible values. Such a field could be replicated in space to compute many features simultaneously.

The construction follows from two ideas: first, the clique potentials of the network are defined such that only two of the many configurations need be considered, the other configurations being penalized by high clique potentials (i.e. low a priori probability). One configuration encodes the decision that the higher-order feature exists, the other that it doesn't exist. The second point is that the energy of the "doesn't exist" configuration is independent of the observation, while the energy of the "exists" configurations improves with the strength of the evidence.

Consider a parameter net for the Hough transform that represents only a single parameterized image segment (e.g. a line segment) and a set of low-level features, (e.g. edges) which vote for it ( Figure 1 left). The variables, labels, and neighborhoods, of the equivalent MRF are defined in the right side of Figure 1 The clique potentials, which depend on the Hough parameters, are shown in the right side of the figure for a single neighborhood of the graph (There are four ways to label this clique.) Unspecified unary potentials are zero. Evidence applies only to the labels $e_i$; it is the likelihood of making a local observation $O_i$:

$$P(O_i \mid e_i) = e^{w_i(f_i - f_{max})} \tag{5}$$

In lemma 1, we show that the configuration $\omega_E = Ee_1e_2\ldots e_n$, has an energy equal to the negated weighted sum of the feature inputs, and configuration $\omega_\theta = \neg Ee_{\neg 1}e_{\neg 2}\ldots \neg e_n$ has a constant energy equal to the negated Hough threshold. Then, in lemma 2, we show that the clique potentials restrict the possible configurations to only these two, so that the network must have its minimum energy in a configuration whose high-level feature has the correct label.

**Lemma 1:**
$U(\omega_E \mid O) = -\sum_{i=1}^{n} w_i f_i$
$U(\omega_\theta \mid O) = -\theta$

**Proof:** The energy contributed by the clique potentials in $\omega_E$ is $\sum_{i=1}^{n} -w_i f_{max}$. Defining $W \equiv \sum_{i=1}^{n} w_i$, this simplifies to $-W f_{max}$.

The evidence also contributes to the energy of $\omega_E$, in the form: $-\sum_{i=1}^{n} \log e_i$. Substituting from 5 into 4 and simplifying gives the total posterior energy of $\omega_E$:

$$U(\omega_E \mid O) = -W f_{max} + W f_{max} - \sum_{i=1}^{n} w_i f_i = -\sum_{i=1}^{n} w_i f_i \qquad (6)$$

The energy of the configuration $\omega_\theta$ does not depend on evidence derived from the Hough features. It has only one clique with a non-zero potential, the unary clique of label $\neg E$. Hence $U(\omega_\theta \mid O) = -\theta.\square$

**Lemma 2:**
$(\forall \omega)(\omega = E \ldots \neg e_k \ldots) \Rightarrow U(\omega \mid O) > U(\omega_E \mid O)$
$(\forall \omega)(\omega = \neg E \ldots e_k \ldots) \Rightarrow U(\omega \mid O) > U(\omega_\theta \mid O)$

**Proof:** For a mixed configuration $\omega = E \ldots \neg e_k \ldots$, changing label $\neg e_k$ to $e_k$ adds energy because of the evidence associated with $e_k$. This is at most $w_i f_{max}$. It also removes energy because of the potential of the clique $E e_k$, which is $-w_i f_{max}$. Because the clique potential $K_2$ from $E \neg e_k$ is also removed, if $K_2 > 0$, then changing this label always reduces the energy.

For a mixed configuration $\omega = \neg E \ldots e_k \ldots$, changing the low-level label $e_k$ to $\neg e_k$ cannot add to the energy contributed by evidence, since $\neg e_k$ has no evidence associated with it. There is no binary clique potential for $\neg E \neg e$, but the potential $K_1$ for clique $\neg E e_k$ is removed. Therefore, again, choosing any $K_1 > 0$ reduces energy and ensures that compatible labels are preferred.$\square$

From lemma 2, there are two configurations that could possibly have minimal posterior energy. From lemma 1, the configuration which represents the existence of the higher-order feature is preferred if and only if the weighted sum of the evidence exceeds threshold, as in the Hough transform.

Often it is desirable to find the mode in a high-level parameter space rather than those elements which surpass a fixed threshold. Finding a single mode is easy to do in a Hough-like MRF; add lateral connections between the *exists* labels of the high-level features to form a winner-take-all network. If the potentials for these cliques are large enough, it is not possible for more than one variable corresponding to a high-level feature to be labeled *exists*.

## 4    BEYOND HOUGH TRANSFORMS: MRF PARAMETER NETS

The essentials of a parameter network are a set of variables representing low-order features, a set of variables representing high-order features, and the appropriate

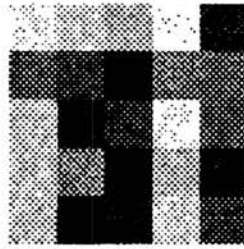

Figure 2: Noisy image data

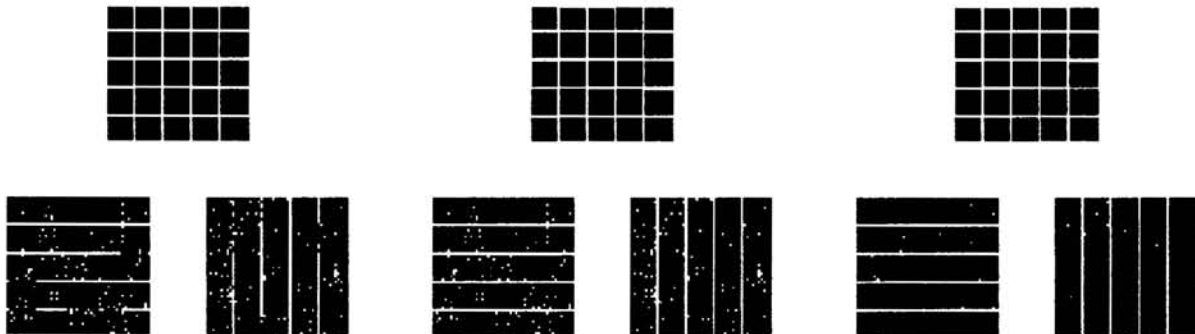

Figure 3: Three parameter-net MRF experiments: white dots in the lower images indicate the decision that a horizontal or vertical local edge is present. Upper images show the horizontal and vertical lines found. The left net is a feedforward Hough transform; the middle net uses positive feedback from lines to edges; the right net uses negative feedback, from non-existing lines to non-existing edges

weighted connections between them. This section explores the characteristics of more "natural" MRF parameter networks, still based on the same variables and connections, but not limited to binary label sets and sum/threshold decision procedures.

## 4.1  EXPERIMENTS WITH FEEDBACK

The Hough transform and its parameter net instantiation are inherently feed-forward. In contrast, all MRFs intrinsically incorporate feedback. We experimented with a network designed to find lines from edges. Horizontal and vertical edge inputs are represented at the low level, and horizontal and vertical lines which span the image at the high level. The input data look like Figure 2. Probabilistic evidence for the low-level edges is generated from pixel data using a model of edge-image formation [Sher, 1987]. The edges vote for compatible lines. In Figure 3, the decision of the feed-forward, Hough transform MRF is shown at the left: edges exist where the local evidence is sufficient; lines exist where enough votes are received.

Keeping the same topology, inputs, and representations in the MRF, we added top-down feedback by changing binary clique potentials so that the existence of a line at the high level is more strongly compatible with the existence of its edges. Missing edges are filled in (middle). By making non-existent lines strongly incompatible with the existence of edges, noisy edges are substantially removed (right). Other MRFs for segmentation [Chou and Brown, 1990; Marroquin, 1985] find collinear edges,

but cannot reason about lines and therefore cannot exploit top-down feedback.

## 4.2  REPRESENTATION AND DECISION MAKING

Both parameter nets and MRFs represent confidence in local hypotheses, but here the MRF framework has intrinsic advantages. MRFs can simultaneously represent independent beliefs for and against the same hypotheses. In an active vision system, which must reason about gathering as well as interpreting evidence, one could extend this to include the label *don't know*, allowing explicit reasoning about the condition in which the local evidence insufficiently supports any decision. MRFs can also express higher-order constraints as more than a set of pairs. The exploitation of appropriate 3-cliques, for example, has been shown to be very useful [Cooper, 1990].

Since the potentials in an MRF are related to local conditional probabilities, there is a principled way to obtain them. Observations can be used to estimate local joint probabilities, which can be converted to the clique potentials defining the prior distribution on the field [Pearl, 1988; Swain, 1990].

Most evidence integration schemes require, in addition to the network topology and parameters, the definition of a decision making process (e.g. thresholding) and a theory of parameter acquisition for that process, which is often ad hoc. To estimate the maximum posterior probability of a MRF, on the other hand, is intrinsically to make a decision among the possibilities embedded in the chosen variables and labels.

The space of possible decisions (interpretations of problem input) is also much richer for MRFs than for parameter networks. For both nets, the nodes for which evidence is available define a $n$-dimensional problem input space. The weights divide this space into regions defined by the one best interpretation (configuration) for all problems in that region. With parameter nets, these regions are separated by planes, since only the sum of the inputs matters. In MRFs, the energy depends on the log-product of the evidence and the sum of the potentials, allowing more general decision surfaces. Non-linear decisions such as AND or XOR are easy to encode, whereas they are impossible for the linear Hough transform.

## 5  CONCLUSION

This paper has shown that parameter networks can be constructed with Markov Random Fields. MRFs can thus bridge representational levels of abstraction in network vision systems. Furthermore, it has been demonstrated that MRFs offer the potential for a significantly more powerful implementation of parameter nets, even if their topological architecture is identical to traditional Hough networks. In short, at least one method is now available for constructing intermediate vision solutions with Markov Random Fields.

It may thus be possible to build entire integrated vision systems with a single well-justified formal framework - Markov Random Fields. Such systems would have a unified representational scheme, constraints and evidence with well-defined semantics, and a single control structure. Furthermore, feedback and feedforward flow of

information, crucial in any complete vision system, is intrinsic to MRFs.

Of course, the task still remains to build a functioning vision system for some domain. In this paper we have said nothing about the definition of specific "features" and the constraints between them that would constitute a useful system. But providing essential tools implemented in a well-defined formal framework is an important step toward building robust, functioning systems.

## Acknowledgements

Support for this research was provided by NSF grant #IRI-9110492 and by Andersen Consulting, through their founding grant to the Institute for the Learning Sciences. Patrick Yuen wrote the MRF simulator that was used in the experiments.

## References

[Ballard, 1984] D.H. Ballard, "Parameter Networks," *Artificial Intelligence*, 22(3):235–267, 1984.

[Bolle *et al.*, 1988] Ruud M. Bolle, Andrea Califano, Rick Kjeldsen, and R.W. Taylor, "Visual Recognition Using Concurrent and Layered Parameter Networks," Technical Report RC-14249, IBM Research Division, T.J. Watson Research Center, Dec 1988.

[Chellapa and Jain, 1991] Rama Chellapa and Anil Jain, editors, *Markov Random Fields: Theory and Application*, Academic Press, 1991.

[Chou and Brown, 1990] Paul B. Chou and Christopher M. Brown, "The Theory and Practice of Bayesian Image Labeling," *International Journal of Computer Vision*, 4:185–210, 1990.

[Cooper, 1990] Paul R. Cooper, "Parallel Structure Recognition with Uncertainty: Coupled Segmentation and Matching," In *Proceedings of the Third International Conference on Computer Vision ICCV '90*, Osaka, Japan, December 1990.

[Geman and Geman, 1984] Stuart Geman and Donald Geman, "Stochastic Relaxation, Gibbs Distributions, and the Bayesian Restoration of Images," *PAMI*, 6(6):721–741, November 1984.

[Hopfield and Tank, 1985] J. J. Hopfield and D. W. Tank, ""Neural" Computation of Decisions in Optimization Problems," *Biological Cybernetics*, 52:141–152, 1985.

[Kirkpatrick *et al.*, 1983] S. Kirkpatrick, C.D. Gelatt, and M.P. Vecchi, "Optimization by Simulated Annealing," *Science*, 220:671–680, 1983.

[Marroquin, 1985] Jose Luis Marroquin, "Probabilistic Solution of Inverse Problems," Technical report, MIT Artificial Intelligence Laboratory, September, 1985.

[Pearl, 1988] Judea Pearl, *Probabalistic Reasoning in Intelligent Systems*, Morgan Kaufman, 1988.

[Sher, 1987] David B. Sher, "A Probabilistic Approach to Low-Level Vision," Technical Report 232, Department of Computer Science, University of Rochester, October 1987.

[Swain, 1990] Michael J. Swain, "Parameter Learning for Markov Random Fields with Highest Confidence First Estimation," Technical Report 350, Dept. of Computer Science, University of Rochester, August 1990.